# Activity Driven Adaptive Stochastic Resonance

**Gregor Wenning and Klaus Obermayer**
Department of Electrical Engineering and Computer Science
Technical University of Berlin
Franklinstr. 28/29, 10587 Berlin
{*grewe,oby*} *@cs.tu-berlin.de*

## Abstract

Cortical neurons might be considered as threshold elements integrating in parallel many excitatory and inhibitory inputs. Due to the apparent variability of cortical spike trains this yields a strongly fluctuating membrane potential, such that threshold crossings are highly irregular. Here we study how a neuron could maximize its sensitivity w.r.t. a relatively small subset of excitatory input. Weak signals embedded in fluctuations is the natural realm of stochastic resonance. The neuron's response is described in a hazard-function approximation applied to an Ornstein-Uhlenbeck process. We analytically derive an optimality criterium and give a learning rule for the adjustment of the membrane fluctuations, such that the sensitivity is maximal exploiting stochastic resonance. We show that adaptation depends only on quantities that could easily be estimated locally (in space and time) by the neuron. The main results are compared with simulations of a biophysically more realistic neuron model.

## 1 Introduction

Energetical considerations [1] and measurements [2] suggest, that sub-threshold inputs, i.e. inputs which on their own are not capable of driving a neuron, play an important role in information processing. This implies that measures must be taken, such that the relevant information which is contained in the inputs is amplified in order to be transmitted. One way to increase the sensitivity of a threshold device is the addition of noise. This phenomenon is called stochastic resonance (see [3] for a review), and has already been investigated and experimentally demonstrated in the context of neural systems ( e.g. [3, 4]). The optimal noise level, however, depends on the distribution of the input signals, hence neurons must adapt their internal noise levels when the statistics of the input is changing. Here we derive and explore an activity dependent learning rule which is intuitive and which only depends on quantities (input and output rates) which a neuron could - in principle - estimate.

The paper is structured as follows. In section 2 we describe the neuron model and we introduce the membrane potential dynamics in its hazard function approximation.

In section 3 we characterize stochastic resonance in this model system and we calculate the optimal noise level as a function of the input and output rates. In section 4 we introduce an activity dependent learning rule for optimally adjusting the internal noise level, demonstrate its usefulness by applying it to the Ornstein-Uhlenbeck neuron and relate the phenomenon of stochastic resonance to its experimentally accessible signature: the adaptation of the neuron's transfer function. Section 5 contains a comparison to the results from a biophysically more realistic neuron model. Section 6, finally, concludes with a brief discussion.

## 2   The abstract Neuron Model

Figure 1 a) shows the basic model setup. A leaky integrate-and-fire neuron receives

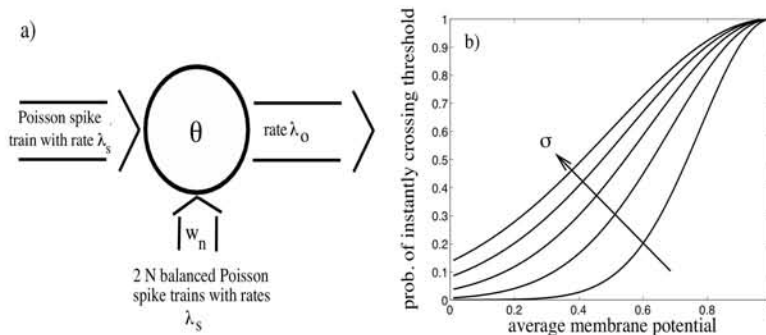

Figure 1: **a)**The basic model setup. For explanation see text. **b)** A family of Arrhenius type hazard functions for different noise levels. 1 corresponds to the threshold $\theta$ and values below 1 are subthreshold.

a "signal" input, which we assume to be a Poisson distributed spike train with a rate $\lambda_s$. The rate $\lambda_s$ is low enough, so that the membrane potential $V$ of the neuron remains sub-threshold and no output spikes are generated. For the following we assume that the information the input and output of the neuron convey is coded by its input and output rates $\lambda_s$ and $\lambda_o$ only. Sensitivity is then increased by adding $2N$ balanced excitatory and inhibitory "noise" inputs ($N$ inputs each) with rates $\lambda_n$ and Poisson distributed spikes. Balanced inputs [5, 6] were chosen, because they do not affect the average membrane potential and allow to separate the effect of decreasing the distance of the neuron's operating point to the threshold potential from the effect of increasing the variance of the noise. Signal and noise inputs are coupled to the neuron via synaptic weights $w_s$ and $w_n$ for the signal and noise inputs. The threshold of the neuron is denoted by $\Theta$. Without loss of generality the membrane time-constant, the neuron's resting potential, and the neuron's threshold are set to one, zero, and one, respectively.

If the total rate $2N\lambda_n$ of incoming spikes on the "noise" channel is large and the individual coupling constants $w_n$ are small, the dynamics of the membrane potential can be approximated by an Ornstein-Uhlenbeck process,

$$dV = -V \; dt + \mu \; dt + \sigma \; dW, \tag{1}$$

where drift $\mu$ and variance $\sigma$ are given by $\mu = w_S \lambda_S$ and $\sigma^2 = w_S^2 \lambda_S + 2N w_N^2 \lambda_N$, and where $dW$ describes a Gaussian noise process with mean zero and variance one [8]. Spike initiation is included by inserting an absorbing boundary with reset. Equation (1) can be solved analytically for special cases [8], but here we opt for

a more versatile approximation (cf. [7]). In this approximation, the probability of crossing the threshold, which is proportional to the instantaneous output rate of the neuron, is described by an effective transfer function. In [7] several transfer functions were compared in their performance, from which we choose an Arrhenius-type function,

$$\lambda_o(t) = c \ \exp\{-\frac{(\theta - x(t))^2}{\sigma^2}\}, \tag{2}$$

where $\theta - x(t)$ is the distance in voltage between the noise free trajectory of the membrane potential $x(t)$ and the threshold $\theta$, $x(t)$ is calculated from eq. (1) without its diffusion term. Note that $x(t)$ is a function of $\lambda_s$, c is a constant. Figure 1 b) shows a family of Arrhenius type transfer functions for different noise levels $\sigma$.

## 3    Stochastic Resonance in an Ornstein-Uhlenbeck Neuron

Several measures can be used to quantify the impact of noise on the quality of signal transmission through threshold devices. A natural choice is the mutual information [9] between the distributions $p(\lambda_s)$ and $p(\lambda_o)$ of input and output rates, which we will discuss in section 4, see also figure 3f. In order to keep the analysis and the derivation of the learning rule simple, however, we first consider a scenario, in which a neuron should distinguish between two sub-threshold input rates $\lambda_s$ and $\lambda_s + \Delta_s$. Optimal distinguishability is achieved if the difference $\Delta_o$ of the corresponding output rates is maximal, i.e. if

$$\Delta_o = f(\lambda_s + \Delta_s) - f(\lambda_s) = \max, \tag{3}$$

where $f$ is the transfer function given by eq. (2). Obviously there is a close connection between these two measures, because increasing both of them leads to an increase in the entropy of $p(\lambda_o)$.

Fig. 2 shows plots of the difference $\Delta_o$ of output rates vs. the level of noise, $\sigma$, for

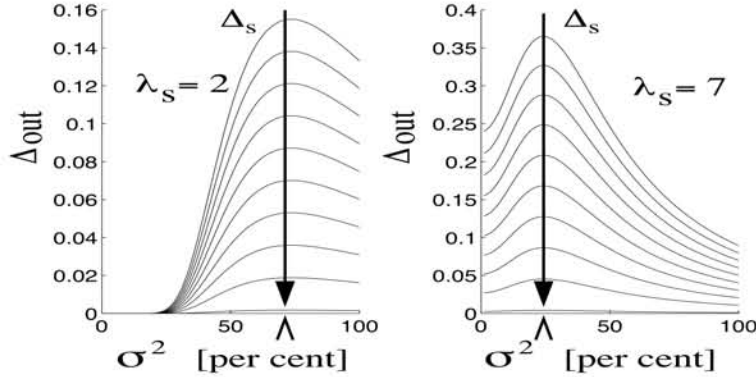

Figure 2: $\Delta_o$ vs. $\sigma^2$ for two different base rates $\lambda_s = 2$ (left) and 7 (right) and 10 different values of $\Delta_s = 0.01, 0.02, ..., 0.1$. $\sigma^2$ is given in per cent of the maximum $\sigma^2 = 2Nw_n^2\lambda_n$. The arrows above the x-axis indicate the position of the maximum according to eq. (3), the arrowheads below the x-axis indicate the optimal value computed using eq. (5) (67% and 25%). Parameters were: $N = 10$, $\lambda_n = 7$, $w_s = 0.1$, and $w_n \in [0, 0.1]$.

different rates $\lambda_s$ and different values of $\Delta_s$. All curves show a clear maximum at a

particular noise level. The optimal noise level increases with decreasing the input rate $\lambda_s$, but is roughly independent of the difference $\Delta_s$ as long as $\Delta_s$ is small. Therefore, one optimal noise level holds even if a neuron has to distinguish several sub-threshold input rates - as long as these rates are clustered around a given base rate $\lambda_s$.

The optimal noise level for constant $\lambda_s$ (stationary states) is given by the condition

$$\frac{d}{d\sigma^2}(f(\lambda_s + \Delta_s) - f(\lambda_s)) = 0 , \qquad (4)$$

where $f$ is given by eq. (2). Equation (4) can be evaluated in the limit of small values of $\Delta_s$ using a Taylor expansion up to the second order. We obtain

$$\sigma_{opt}^2 = 2(1 - w_s\lambda_s)^2 \qquad (5)$$

if the main part of the variance of the membrane potential is a result of the balanced input, i.e. if $\sigma^2 \approx 2Nw_N^2\lambda_N$ (cf. eq. (1)). Since $\sigma_{opt}^2 = -\frac{(1-w_s\lambda_s)^2}{\log(\lambda_o/c)}$, eq. (2), eq. (5) is equivalent to $1 + 2 \log(\frac{\lambda_o(\lambda_s;\sigma^2)}{c}) = 0$. This shows that the optimal noise level depends either only on $\lambda_s$ or on $\lambda_o(\lambda_s;\sigma^2)$, both are quantities which are locally available at the cell.

## 4   Adaptive Stochastic Resonance

We now consider the case, that a neuron needs to adapt its internal noise level because the base input rate $\lambda_s$ changes. A simple learning rule which converges to the optimal noise level is given by

$$\Delta\sigma^2 = -\epsilon \, \log(\frac{\sigma^2}{\sigma_{opt}^2}), \qquad (6)$$

where the learning parameter $\epsilon$ determines the time-scale of adaptation. Inserting the corresponding expressions for the actual and the optimal variance we obtain a learning rule for the weights $w_n$,

$$\Delta w_n = -\epsilon \, \log(\frac{2N\lambda_n w_n^2}{2(1 - w_s\lambda_s)^2}). \qquad (7)$$

Note, that equivalent learning rules (in the sense of eq. (6)) can be formulated for the number $N$ of the noise inputs and for their rates $\lambda_n$ as well. The r.h.s. of eqs. (6) and (7) depend only on quantities which are locally available at the neuron.

Fig. 3ab shows the stochastic adaptation of the noise level, using eq. (7), to randomly distributed $\lambda_s$ which are clustered around a base rate.

Fig. 3c-f shows an application of the learning rule, eq. (7) to an Ornstein-Uhlenbeck neuron whose noise level needs to adapt to three different base input rates. The figure shows the base input rate $\lambda_s$ (Fig. 3a). In fig. 3b the adaptation of $w_n$ according to eq. (7) is shown (solid line), for comparison the $w_n$ which maximizes eq. (3) is also displayed (dashed dotted line). Mutual information was calculated between a distribution of randomly chosen input rates which are clustered around the base rate $\lambda_s$. The $w_n$ that maximizes mutual Information between input and output rates is displayed in fig. 3d (dashed line). Fig. 3e shows the ratio $\Delta_o/\Delta_s$ computed by using eq. (3) and the $w_n$ calculated with eq. (8) (dashed dotted line) and the same ratio for the quadratic approximation. Fig. 3f shows the mutual information between the input and output rates as a function of the changing $w_n$.

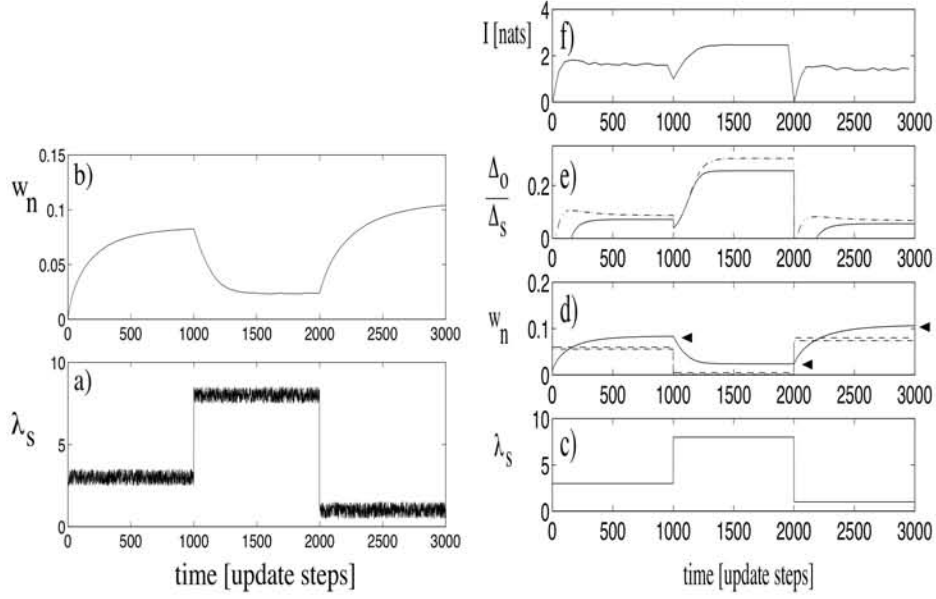

Figure 3: **a)** Input rates $\lambda_s$ are evenly distributed around a base rate with width 0.5, in each time step one $\lambda_s$ is presented. **b)** Application of the learning rule eq. (7) to the rates shown in a). Adaptation of the noise level to three different input base rates $\lambda_s$. **c)** The three base rates $\lambda_s$. **d)** $w_n$ as a function of time according to eq. (7) (solid line), the optimal $w_n$ that maximizes eq. (3) (dashed dotted line) and the optimal $w_n$ that maximizes the mutual information between the input and output rates (dashed). The optimal values of $w_n$ as the quadratic approximation, eq. (5) yield are indicated by the black arrows. **e)** The ratio $\Delta_o/\Delta_s$ computed from eq. (3) (dashed dotted line) and the quadratic approximation (solid line). **f)** Mutual information between input and output rates as a function of base rate and changing synaptic coupling constant $w_n$. For calculating the mutual information the input rates were chosen randomly from the interval $[\lambda_s - 0.25, \lambda_s + 0.25]$ in each time step. Parameters as in fig. 2.

The figure shows, that the learning rule, eq. (7) in the quadratic approximation leads to values for $\sigma$ which are near-optimal, and that optimizing the difference of output rates leads to results similar to the optimization of the mutual information.

## 5    Conductance based Model Neuron

To check if and how the results from the abstract model carry over to a biophysically mode realistic one we explore a modified Hodgkin-Huxley point neuron with an additional A-Current (a slow potassium current) as in [11]. The dynamics of the membrane potential $V$ is described by the following equation

$$
\begin{aligned}
C\frac{dV}{dt} &= -g_L(V(t) - E_L) - \bar{g}_{Na}m_\infty^3 h(t)(V - E_{Na}) \\
&\quad -\bar{g}_K n(t)^4(V - E_K) - \bar{g}_A a_\infty^3 b(t)(V - E_K) \\
&\quad + I_{syn} + I_{app},
\end{aligned} \tag{8}
$$

the parameters can be found in the appendix. All parameters are kept fixed through-

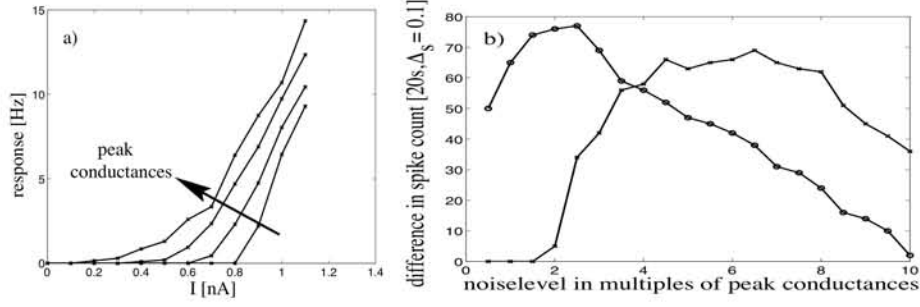

Figure 4: **a)** Transfer function for the conductance based model neuron with additional balanced input, $a = 1, 2, 3, 4$ **b)** Demonstration of SR for the conductance based model neuron. The plot shows the resonance for two different base currents $I_{app} = 0.7$ and $I_{app} = 0.2$ and $a \in [0, 10]$.

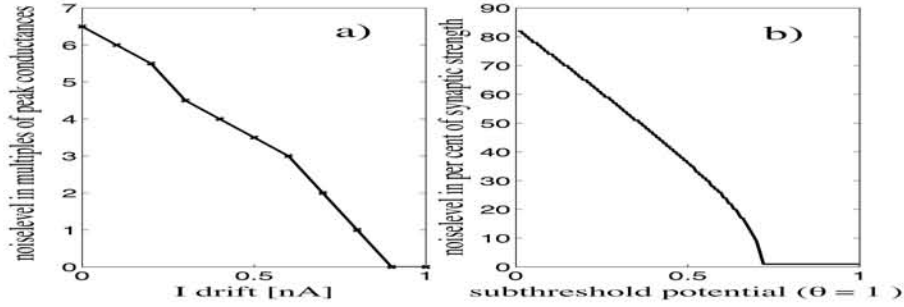

Figure 5: **a)** Optimal noise-level as a function of the base current in the conductance based model. **b)** Optimal noise-level as a function of the noise-free membrane potential in the abstract model.

out all shown data. As balanced input we choose an excitatory Poisson spike train with rate $\lambda_{ne} = 1750$ Hz and an inhibitory spike train with rate $\lambda_{ni} = 750$ Hz. These spike trains are coupled to the neuron via synapses resulting in a synaptic current as in [12]

$$I_{syn} = g_e(V(t) - E_e) + g_i(V(t) - E_i)). \tag{9}$$

Every time a spike arrives at the synapse the conductance is increased by its peak conductance $\bar{g}_{e,i}$ and decreases afterwards exponentially like $exp\{-\frac{t}{\tau_{e,i}}\}$. The corresponding parameters are $\bar{g}_e = a * 0.02 * g_L$, $\bar{g}_i = a * 0.0615 * g_L$. The common factor $a$ is varied in the simulations and adjusts the height of the peak conductances, $g_L$ is the leak conductance given above. Excitatory and inhibitory input are called balanced if the impact of a spike-train at threshold is the same for excitation and inhibition

$$\tilde{\tau}_e \bar{g}_e \lambda_{ne}(E_e - \theta) = -\tilde{\tau}_i \bar{g}_i \lambda_{ni}(E_i - \theta) \tag{10}$$

with $\tilde{\tau}_{e,i} = \frac{1}{\bar{g}_{e,i}} \int_0^\infty g_{e,i}(t)dt$. Note that the factor $a$ does cancel in eq. (10).

Fig. 4a displays transfer functions in the conductance based setting with balanced input. A family of functions with varying peak conductances for the balanced input is drawn.

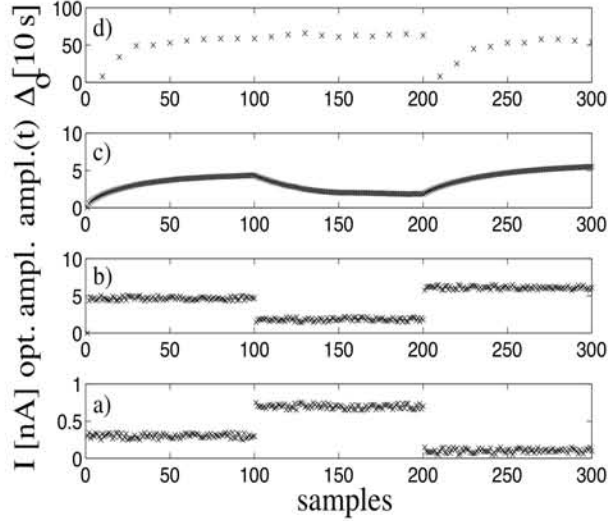

Figure 6: Adaptive SR in the conductance based model. **a)** Currents drawn from a uniform distribution of width $0.2\ nA$ centered around base currents of 3, 8, 1 $nA$ respectively. **b)** Optimal noise-level in terms of $a$. Optimality refers to a semi-linear fit to the data of fig. 5a. **c)** adapting the peak conductances using $a$ in a learning rule like eq. (8). **d)** Difference in spike count, for base currents $I \pm 0.1\ nA$ and using $a$ as specified in c).

For studying SR in the conductance based framework, we apply the same paradigm as in the abstract model. Given a certain average membrane potential, which is adjusted via injecting a current $I$ (in $nA$), we calculate the difference in the output rate given a certain difference in the average membrane potential (mediated via the injected current) $I \pm \Delta I$. A demonstration of stochastic resonance in the conductance based neuron can be seen in fig. 4b. In fig. 5a the optimal noise-level, in terms of multiples $a$ of the peak conductances, is plotted versus all currents that yield a sub-threshold membrane voltage. For comparison we give the corresponding relationship for the abstract model in fig. 5b.

Fig. 6 shows the performance of the conductance based model using a learning rule like eq. (7). Since we do not have an analytically derived expression for $\sigma_{opt}$ in the conductance based case, the relation $\sigma_{opt}(I)$, necessary for using eq. (7), corresponds to a semi-linear fit to the $(a_{opt}, I)$ relation in fig. 5a.

# 6   Conclusion and future directions

In our contribution we have shown, that a simple and activity driven learning rule can be given for the adaptation of the optimal noise level in a stochastic resonance setting. The results from the abstract framework are compared with results from a conductance based model neuron. A biological plausible mechanism for implementing adaptive stochastic resonance in conductance based neurons is currently under investigation.

## Acknowledgments

Supported by: Wellcome Trust (061113/Z/00)

## Appendix: Parameters for the conductance based model neuron

somatic conductances/ion-channel properties: $C_m = 1.0 \; \frac{\mu F}{cm^2}, g_L = 0.05 \; \frac{mS}{cm^2}, g_{Na} = 100 \; \frac{mS}{cm^2}, g_K = 40 \; \frac{mS}{cm^2}, g_A = 20 \; \frac{mS}{cm^2}, E_L = -65 \; mV, E_{Na} = 55 \; mV, E_K = -80 \; mV, \tau_A = 20 \; ms,$
synaptic coupling: $E_e = 0 \; mV, \; E_i = -80 \; mV, \; \tau_e = 5 \; ms, \; \tau_i = 10 \; ms,$
spike initiation: $\frac{dh}{dt} = \frac{h_\infty - h}{\tau_h}, \frac{dn}{dt} = \frac{n_\infty - n}{\tau_n}, \frac{db}{dt} = \frac{b_\infty - b}{\tau_A},$
$m_\infty = \frac{\alpha_m}{\alpha_m + \beta_m}, \; \alpha_m = -0.1(V + 30)/(exp(-0.1(V + 30)) - 1), \; \beta_m = 4exp(-(V + 55)/18),$
$h_\infty = \frac{\alpha_h}{\alpha_h + \beta_h}, \; \alpha_h = 0.07exp(-(V + 44)/20), \beta_h = 1/(exp(-0.1(V + 14)) + 1),$
$n_\infty = \frac{\alpha_n}{\alpha_n + \beta_n}, \; \alpha_n = -0.01(V + 34)/(exp(-0.1(V + 34)) - 1), \; \beta_n = 0.125exp(-(V + 44)/80)$
$a_\infty = 1/(exp(-(V + 50)/20) + 1), b_\infty = 1/(exp((V + 80)/6) + 1),$
$\tau_h = t_n/(\alpha_h + \beta_h), \tau_n = t_n/(\alpha_n + \beta_n), t_n = 0.1$

# References

[1] S. B. Laughlin, R.R. de Ruyter van Steveninck and J.C. Anderson, *The metabolic cost of neural information*, Nature Neuroscience, **1**(1), 1998, p.36-41

[2] J. S. Anderson, I. Lampl, D. C. Gillespie and D. Ferster, *The Contribution of Noise to Contrast Invariance of Orientation Tuning in Cat Visual Cortex*, Science, **290**, 2000, p.1968-1972

[3] L. Gammaitoni, P. Hänggi, P. Jung and F. Marchesoni, *Stochastic Resonance* Reviews of modern Physics, **70**(1), 1998, p.223-287

[4] D. F. Russel, L. A. Wilkens and F. Moss, *Use of behavioral stochastic resonance by paddle fish for feeding*, Nature, **402**, 1999, p.291-294

[5] M. N. Shadlen, and W. T. Newsome, *The Variable Discharge of Cortical Neurons: Implications for Connectivity, Computation, and Information Coding*, The Journal of Neuroscience, **18**(10), 1998, p.3870-3896

[6] M.V. Tsodyks and T. Sejnowski, *Rapid state switching in balanced cortical network models*, Network: Computation in Neural Systems, **6**, 1995, p.111-124

[7] H. E. Plesser and W. Gerstner, *Noise in Integrate-and-Fire Neurons: From Stochastic Input to Escape Rates*, Neural Computation, **12**, 2000, p.367-384

[8] H. C. Tuckwell, *Introduction to theoretical neurobiology: volume 2 nonlinear and stochastic theories*, Cambridge University Press, 1998

[9] T. M. Cover and J. A. Thomas, *Elements of Information Theory*, Wiley Series in Telecommunications, 1991, 2nd edition

[10] A.R. Bulsara, and A. Zador, *Threshold detection of wideband signals: a noise-induced maximum in the mutual information.*, PRE, **54**(3), 1996, R2185-2188

[11] O. Shriki, D. Hansel and H. Sompolinsky, *Modeling neuronal networks in cortex by rate models using the current-frequency response properties of cortical cells*, Soc. Neurosci. Abstr., **24**, p.143, 1998

[12] E. Salinas and T.J. Sejnowski, *Impact of Correlated Synaptic Input on Output Firing Rate and Variability in Simple Neuronal Models* J. Neurosci. **20**, 2000, p.6193-6209
